# Bayesian Backprop in Action: Pruning, Committees, Error Bars and an Application to Spectroscopy

**Hans Henrik Thodberg**
Danish Meat Research Institute
Maglegaardsvej 2, DK-4000 Roskilde
thodberg@nn.meatre.dk

## Abstract

MacKay's Bayesian framework for backpropagation is conceptually appealing as well as practical. It automatically adjusts the weight decay parameters during training, and computes the evidence for each trained network. The evidence is proportional to our belief in the model. The networks with highest evidence turn out to generalise well. In this paper, the framework is extended to pruned nets, leading to an Ockham Factor for "tuning the architecture to the data". A committee of networks, selected by their high evidence, is a natural Bayesian construction. The evidence of a committee is computed. The framework is illustrated on real-world data from a near infrared spectrometer used to determine the fat content in minced meat. Error bars are computed, including the contribution from the dissent of the committee members.

## 1 THE OCKHAM FACTOR

William of Ockham's (1285-1349) principle of economy in explanations, can be formulated as follows:

> *If several theories account for a phenomenon we should prefer the simplest which describes the data sufficiently well.*

The principle states that a model has two virtues: simplicity and goodness of fit. But what is the meaning of "sufficiently well" - i.e. what is the optimal trade-off between the two virtues? With Bayesian model comparison we can deduce this trade-off.

We express our belief in a model as its probability given the data, and use Bayes' formula:

$$P(\mathcal{H}\,|\,D) = \frac{P(D\,|\,\mathcal{H})P(\mathcal{H})}{P(D)} \tag{1}$$

We assume that the prior belief $P(\mathcal{H})$ is the same for all models, so we can compare models by comparing $P(D\,|\,\mathcal{H})$ which is called *the evidence for* $\mathcal{H}$, and acts as a quality measure in model comparison.

Assume that the model has a single tunable parameter $w$ with a prior range $\Delta w_{\mathrm{prior}}$ so that $P(w\,|\,\mathcal{H}) = 1/\Delta w_{\mathrm{prior}}$. The most probable (or maximum posterior) value $w_{\mathrm{MP}}$ of the parameter $w$ is given by the maximum of

$$P(w\,|\,D,\mathcal{H}) = \frac{P(D\,|\,w,\mathcal{H})P(w\,|\,\mathcal{H})}{P(D\,|\,\mathcal{H})} \tag{2}$$

The width of this distribution is denoted $\Delta w_{\mathrm{posterior}}$. The evidence $P(D\,|\,\mathcal{H})$ is obtained by integrating over the posterior $w$ distribution and approximating the integral:

$$P(D|\mathcal{H}) = \int P(D\,|\,w,\mathcal{H})P(w\,|\,\mathcal{H})dw \tag{3}$$

$$= P(D\,|\,w_{\mathrm{MP}},\mathcal{H})\frac{\Delta w_{\mathrm{posterior}}}{\Delta w_{\mathrm{prior}}} \tag{4}$$

$$\mathrm{Evidence} = \mathrm{Likelihood} \times \mathrm{OckhamFactor} \tag{5}$$

The evidence for the model is the product of two factors:

- The best fit likelihood, i.e. the probability of the data given the model and the tuned parameters. It measures how well the *tuned* model fits the data.

- The integrated probability of the tuned model parameters with their uncertainties, i.e. the collapse of the available parameter space when the data is taken into account. This factor is small when the model has many parameters or when some parameters must be tuned very accurately to fit the data. It is called *the Ockham Factor* since it is large when the model is simple.

By optimizing the modelling through the evidence framework we can avoid the overfitting problem as well as the equally important "underfitting" problem.

## 2    THE FOUR LEVELS OF INFERENCE

In 1991-92 MacKay presented a comprehensive and detailed framework for combining backpropagation neural networks with Bayesian statistics (MacKay, 1992). He outlined four levels of inference which applies for instance to a regression problem where we have a training set and want to make predictions for new data:

**Level 1** Make predictions including error bars for new input data.

**Level 2** Estimate the weight parameters and their uncertainties.

**Level 3** Estimate the scale parameters (the weight decay parameters and the noise scale parameter) and their uncertainties.

**Level 4** Select the network architecture and for that architecture select one of the **w**-minima. Optionally select a committee to reflect the uncertainty on this level.

Level 1 is the typical goal in an application. But to make predictions we have to do some modelling, so at level 2 we pick a net and some weight decay parameters and train the net for a while. But the weight decay parameters were picked rather arbitrarily, so on level 3 we set them to their inferred maximum posterior (MP) value. We alternate between level 2 and 3 until the network has converged. This is still not the end, because also the network architecture was picked rather arbitrarily. Hence level 2 and 3 are repeated for other architectures and the evidences of these are computed on level 4. (Pruning makes level 4 more complicated, see section 6).

When we make inference on each of these levels, there are uncertainties which are described by the posterior distributions of the parameters which are inferred. The uncertainty on level 2 is described by the Hessian (the second derivative of the net cost function with respect to the weights). The uncertainty on level 3 is negligible if the number of weight decays parameters is small compared to the number of weights. The uncertainty on level 4 is described by the committee of networks with highest evidence within some margin (discussed below).

The uncertainties are used for two purposes. Firstly they give rise to error bars on the predictions on level 1. And secondly the posterior uncertainty divided by the prior uncertainty (the Ockham Factor) enters the evidence.

MacKay's approach differs in two respects from other Bayesian approaches to neural nets:

- It assumes the Gaussian approximation to the posterior weight distribution. In contrast, the Monte Carlo approach of (Neal, 1992) does not suffer from this limitation.

- It determines maximum posterior values of the weight decay parameters, rather than integrating them out as done in (Buntine and Weigend, 1991).

It is difficult to justify these choices in general. The Gaussian approximation is believed to be good when there are at least 3 training examples per weight (MacKay, 1992). The use of MP weight decay parameters is the superior method when there are ill-defined parameters, as there usually is in neural networks, where some weights are typically poorly defined by the data (MacKay, 1993).

## 3   BAYESIAN NEURAL NETWORKS

The training set $D$ consists of $N$ cases of the form $(\mathbf{x}, t)$. We model $t$ as a function of $\mathbf{x}$, $t = y(\mathbf{x}) + \nu$, where $\nu$ is Gaussian noise and $y(\mathbf{x})$ is computed by a neural

network $\mathcal{H}$ with weights $\mathbf{w}$. The noise scale is a free parameter $\beta = 1/\sigma_\nu^2$. The probability of the data (the likelihood) is

$$P(D|\mathbf{w},\beta,\mathcal{H}) \quad \propto \quad \exp(-\beta E_D) \tag{6}$$

$$E_D \quad \equiv \quad \tfrac{1}{2}\sum(y-t)^2 \tag{7}$$

where the sum extends over the $N$ cases.

In Bayesian modelling we must specify the prior distribution of the model parameters. The model contains $k$ adjustable parameters $\mathbf{w}$, called weights, which are in general split into several groups, for instance one per layer of the net. Here we consider the case with all weights in one group. The general case is described in (MacKay, 1992) and in more details in (Thodberg, 1993). The prior of the weights $\mathbf{w}$ is

$$P(\mathbf{w}|\beta,\xi,\mathcal{H}) \quad \propto \quad \exp(-\beta\xi E_W) \tag{8}$$

$$E_W \quad \equiv \quad \tfrac{1}{2}\sum w^2 \tag{9}$$

$\beta$ and $\xi$ are called *the scales* of the model and are free parameters determined by the data.

The most probable values of the weights given the data, some values of the scales (to be determined later) and the model, is given by the maximum of

$$P(\mathbf{w}|D,\beta,\xi,\mathcal{H}) \quad = \quad \frac{P(D|\mathbf{w},\beta,\xi,\mathcal{H})P(\mathbf{w}|\beta,\xi,\mathcal{H})}{P(D|\beta,\xi,\mathcal{H})} \propto \exp(-\beta C) \tag{10}$$

$$C \quad \equiv \quad E_D + \xi E_W \tag{11}$$

So the maximum posterior weights according to the probabilistic interpretation are identical to the weights obtained by minimising the familiar cost function $C$ with weight decay parameter $\xi$. This is the well–known Bayesian account for weight decay.

## 4   MACKAY'S FORMULAE

The single most useful result of MacKay's analysis is a simple formula for the MP value of the weight decay parameter

$$\xi_{\mathrm{MP}} = \frac{E_D}{E_W}\frac{\gamma}{N-\gamma} \tag{12}$$

where $\gamma$ is the number of well–determined parameters which can be approximated by the actual number of parameters $k$, or computed more accurately from the eigenvalues $\lambda_i$ of the Hessian $\nabla\nabla E_D$:

$$\gamma \equiv \sum_{i=1}^{k} \frac{\lambda_i}{\lambda_i + \xi_{\mathrm{MP}}} \tag{13}$$

The MP value of the noise scale is $\beta_{\mathrm{MP}} = N/(2C)$.

The evidence for a neural network $\mathcal{H}$ is, as in section 1, obtained by integration over the posterior distribution of the inferred parameters, which gives raise to the Ockham Factors:

$$\text{Ev}(\mathcal{H}) \equiv \log P(D|\mathcal{H}) = -\frac{N-\gamma}{2} - \frac{N}{2}\log\frac{4\pi C}{N}$$
$$+ \log\text{Ock}(\mathbf{w}) + \log\text{Ock}(\beta) + \log\text{Ock}(\xi) \qquad (14)$$

$$\log\text{Ock}(\mathbf{w}) = \frac{1}{2}\sum_{i=1}^{k}\log\frac{\xi_{\text{MP}}}{\xi_{\text{MP}}+\lambda_i} - \frac{\gamma}{2} + \log h! + h\log 2 \qquad (15)$$

$$\text{Ock}(\beta) = \frac{\sqrt{4\pi/(N-\gamma)}}{\log\Omega} \quad \text{Ock}(\xi) = \frac{\sqrt{4\pi/\gamma}}{\log\Omega} \qquad (16)$$

The first line in (14) is the log likelihood. The Ockham Factor for the weights $\text{Ock}(\mathbf{w})$ is small when the eigenvalues $\lambda_i$ of the Hessian are large, corresponding to well-determined weights. $\Omega$ is the prior range of the scales and is set (subjectively) to $10^3$.

The expression (15) (valid for a network with a single hidden layer) contains a symmetry factor $h!2^h$. This is because the posterior volume must include all $\mathbf{w}$ configurations which are equivalent to the particular one. The hidden units can be permuted, giving a factor $h!$ more posterior volume. And the sign of the weights to and from every hidden unit can be changed giving $2^h$ times more posterior volume.

## 5   COMMITTEES

For a given data set we usually train several networks with different numbers of hidden units and different initial weights. Several of these networks have evidence near or at the maximal value, but the networks differ in their predictions. The different solutions are interpreted as components of the posterior distribution and the correct Bayesian answer is obtained by averaging the predictions over the solutions, weighted by their posterior probabilities, i.e. their evidences. However, the evidence is not accurately determined, primarily due to the Gaussian approximation. This means that instead of weighting with $\text{Ev}(\mathcal{H})$ we should use the weight $\exp(\log\text{Ev}/\Delta(\log\text{Ev}))$, where $\Delta(\log\text{Ev})$ is the total uncertainty in the evaluation of $\log\text{Ev}$. As an approximation to this, we define the committee as the models with evidence larger than $\log\text{Ev}_{\text{max}} - \Delta\log\text{Ev}$, where $\text{Ev}_{\text{max}}$ is the largest evidence obtained, and all members enter with the same weight.

To compute the evidence $\text{Ev}(\mathcal{C})$ of the committee, we assume for simplicity that all networks in the committee $\mathcal{C}$ share the same architecture. Let $N_{\mathcal{C}}$ be the number of truly different solutions in the committee. Of course, we count symmetric realisations only once. The posterior volume i.e. the Ockham Factor for the weights is now $N_{\mathcal{C}}$ times larger. This renders the committee more probable – it has a larger evidence:

$$\log\text{Ev}(\mathcal{C}) = \log N_{\mathcal{C}} + \log\text{Ev}(\mathcal{H}) \qquad (17)$$

where $\log\text{Ev}(\mathcal{H})$ denotes the average log evidence of the members. Since the evidence is correlated with the generalisation error, we expect the committee to generalise better than the committee members.

# 6  PRUNING

We now extend the Bayesian framework to networks which are pruned to adjust the architecture to the particular problem. This extends the fourth level of inference.

At first sight, the factor $h!$ in the Ockham Factor for the weights in a sparsely connected network appears to be lost, since the network is (in general) not symmetric with respect to permutations of the hidden units. However, the symmetry reappears because for every sparsely connected network with tuned weights there are $h!$ other equivalent network architectures obtained by permuting the hidden units. So the factor $h!$ remains. If this argument is not found compelling, it can be viewed as an assumption.

If the data are used to select the architecture, which is the case in pruning designed to minimise the cost function, an additional Ockham Factor must be included. With one output unit, only the input–to–hidden layer is sparsely connected, so consider only these connections. Attach a binary pruning parameter to each of the $m$ potential connections. A sparsely connected architecture is described by the values of the pruning parameters. The prior probability of a connection to be present is described by a hyperparameter $\phi$ which is determined from the data i.e. it is set to the fraction of connections remaining after pruning (notice the analogy between $\phi$ and a weight decay parameter). A non–pruned connection gives an Ockham Factor $\phi$ and a pruned $1 - \phi$, assuming the data to be certain about the architecture. The Ockham Factors for the pruning parameters is therefore

$$\log \mathrm{Ock(pruning)} = m(\phi_{\mathrm{MP}} \log \phi_{\mathrm{MP}} + (1 - \phi_{\mathrm{MP}}) \log(1 - \phi_{\mathrm{MP}})) \qquad (18)$$

The tuning of the meta-parameter to the data gives an Ockham factor $\mathrm{Ock}(\phi) \approx \sqrt{2/m}$, which is rather negligible.

From a minimum description length perspective (18) reflects the extra information needed to describe the topology of a pruned net relative to a fully connected net. It acts like a barrier towards pruning. Pruning is favoured only if the negative contribution $\log \mathrm{Ock(pruning)}$ is compensated by an increase in for instance $\log \mathrm{Ock}(\mathbf{w})$.

# 7  APPLICATION TO SPECTROSCOPY

Bayesian Backprop is used in a real-life application from the meat industry. The data were recorded by a Tecator near-infrared spectrometer which measures the spectrum of light transmitted through samples of minced pork meat. The absorbance spectrum has 100 channels in the region 850-1050 nm. We want to calibrate the spectrometer to determine the fat content. The first 10 principal components of the spectra are used as input to a neural network.

Three weight decay parameters are used: one for the weights and biases of the hidden layer, one for the connections from the hidden to the output layer, and one for the direct connections from the inputs to the output as well as the output bias.

The relation between test error and log evidence is shown in figure 1. The test error is given as *standard error of prediction* (SEP), i.e. the root mean square error. The 12 networks with 3 hidden units and evidence larger than −270 are selected for a

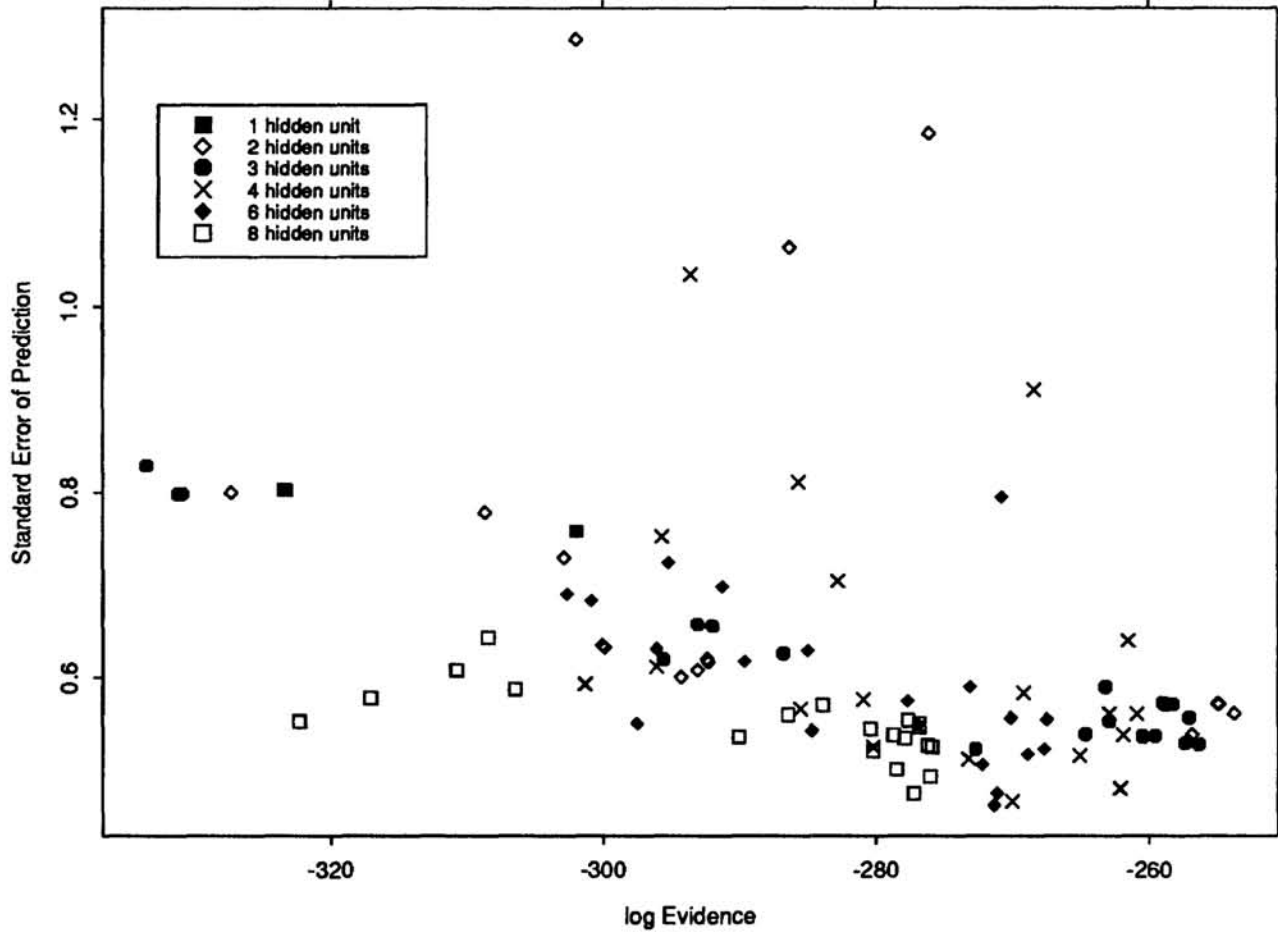

Figure 1: The test error as a function of the log evidence for networks trained on the spectroscopic data. High evidence implies low test error.

committee. The committee average gives 6% lower SEP than the members do on average, and 21% lower SEP than a non-Bayesian analysis using early stopping (see Thodberg, 1993).

Pruning is applied to the networks with 6 hidden units. The evidence decreases slightly, i.e. Ock(pruning) dominates. Also the SEP is slightly worse. So the evidence correctly suggests that pruning is not useful for this problem. [1]

The Bayesian error bars are illustrated for the spectroscopic data in figure 2. We study the model predictions on the line through input space defined by the second principal component axis, i.e. the second input is varied while all other inputs are zero. The total prediction variance for a new datum $\mathbf{x}$ is

$$\sigma_{\text{total}}(\mathbf{x})^2 = \sigma_\nu^2 + \sigma_{\text{WU}}(\mathbf{x})^2 + \sigma_{\text{CU}}(\mathbf{x})^2 \qquad (19)$$

where $\sigma_{\text{WU}}$ comes from the weight uncertainties (level 2) and $\sigma_{\text{CU}}$ from the committee dissent (level 4).

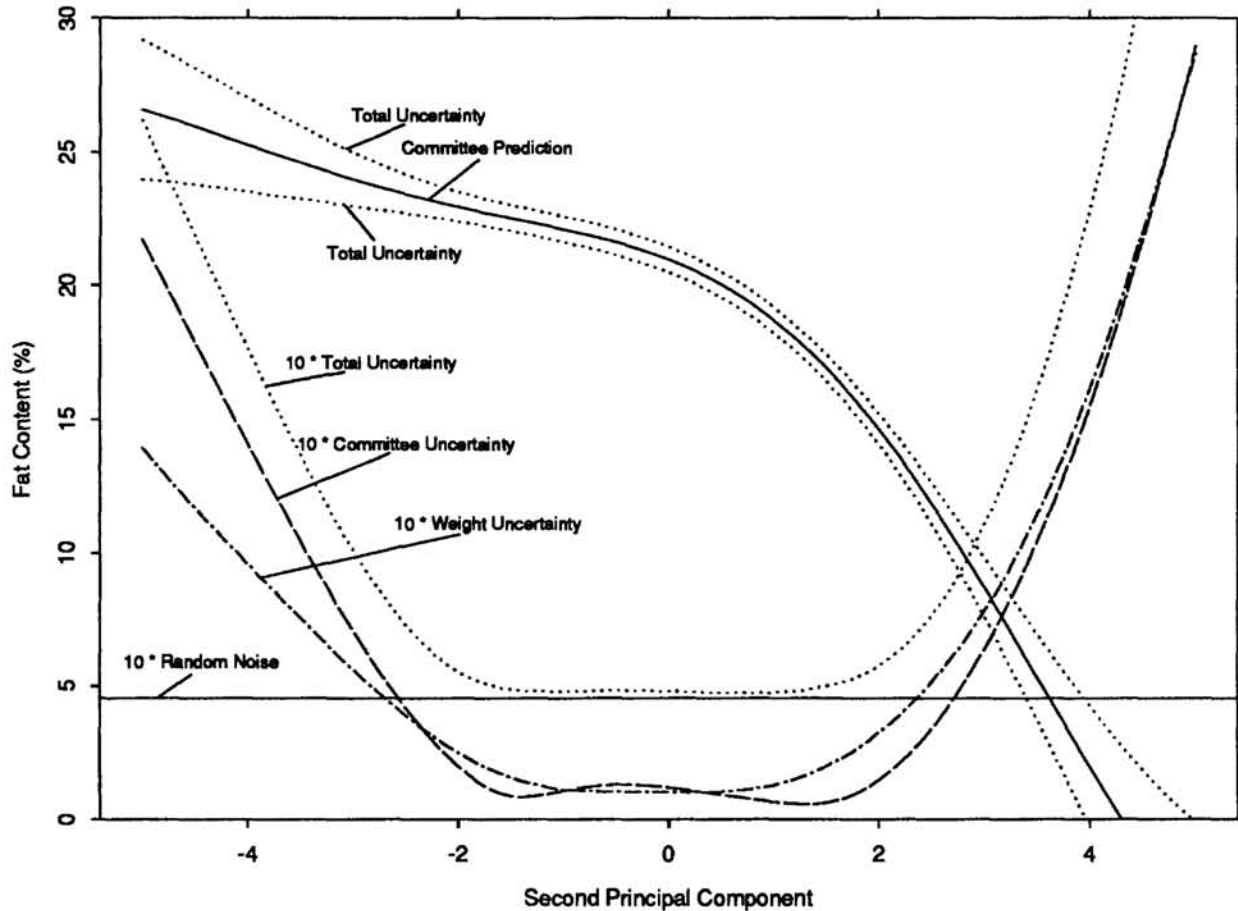

Figure 2: Prediction of the fat content as a function of the second principal component $p_2$ of the NIR spectrum. 95% of the training data have $|p_2| < 2$. The total error bars are indicated by a "1 sigma" band with the dotted lines. The total standard errors $\sigma_{\text{total}}(\mathbf{x})$ and the standard errors of its contributions ($\sigma_\nu$, $\sigma_{\text{WU}}(\mathbf{x})$ and $\sigma_{\text{CU}}(\mathbf{x})$) are shown separately, multiplied by a factor of 10.

## Footnotes

[1] For artificial data generated by a sparsely connected network the evidence correctly points to pruned nets as better models (see Thodberg, 1993).

## References

W.L.Buntine and A.S.Weigend, "Bayesian Back-Propagation", *Complex Systems* **5**, (1991) 603-643.

R.M.Neal, "Bayesian Learning via Stochastic Dynamics", *Neural Information Processing Systems, Vol.5*, ed. C.L.Giles, S.J.Hanson and J.D.Cowan (Morgan Kaufmann, San Mateo, 1993)

D.J.C.MacKay, "A Practical Bayesian Framework for Backpropagation Networks" *Neural Comp.* **4** (1992) 448-472.

D.J.C.MacKay, paper on Bayesian hyperparameters, in preparation 1993.

H.H.Thodberg, "A Review of Bayesian Backprop with an Application to Near Infrared Spectroscopy" and "A Bayesian Approach to Pruning of Neural Networks", submitted to *IEEE Transactions of Neural Networks* 1993 (in /pub/neuroprose/thodberg.ace-of-bayes*.ps.Z on archive.cis.ohio-state.edu).
